# Variance Penalizing AdaBoost

**Pannagadatta K. Shivaswamy**
Department of Computer Science
Cornell University, Ithaca NY
pannaga@cs.cornell.edu

**Tony Jebara**
Department of Compter Science
Columbia University, New York NY
jebara@cs.columbia.edu

## Abstract

This paper proposes a novel boosting algorithm called VadaBoost which is motivated by recent empirical Bernstein bounds. VadaBoost iteratively minimizes a cost function that balances the sample mean and the sample variance of the exponential loss. Each step of the proposed algorithm minimizes the cost efficiently by providing weighted data to a weak learner rather than requiring a brute force evaluation of all possible weak learners. Thus, the proposed algorithm solves a key limitation of previous empirical Bernstein boosting methods which required brute force enumeration of all possible weak learners. Experimental results confirm that the new algorithm achieves the performance improvements of EBBoost yet goes beyond decision stumps to handle any weak learner. Significant performance gains are obtained over AdaBoost for arbitrary weak learners including decision trees (CART).

## 1  Introduction

Many machine learning algorithms implement empirical risk minimization or a regularized variant of it. For example, the popular AdaBoost [4] algorithm minimizes exponential loss on the training examples. Similarly, the support vector machine [11] minimizes hinge loss on the training examples. The convexity of these losses is helpful for computational as well as generalization reasons [2]. The goal of most learning problems, however, is not to obtain a function that performs well on training data, but rather to estimate a function (using training data) that performs well on future unseen test data. Therefore, empirical risk minimization on the training set is often performed while regularizing the complexity of the function classes being explored. The rationale behind this regularization approach is that it ensures that the empirical risk converges (uniformly) to the true unknown risk. Various concentration inequalities formalize the rate of convergence in terms of the function class complexity and the number of samples.

A key tool in obtaining such concentration inequalities is Hoeffding's inequality which relates the empirical mean of a bounded random variable to its true mean. Bernstein's and Bennett's inequalities relate the true mean of a random variable to the empirical mean but also incorporate the true variance of the random variable. If the true variance of a random variable is small, these bounds can be significantly tighter than Hoeffding's bound. Recently, there have been empirical counterparts of Bernstein's inequality [1, 5]; these bounds incorporate the empirical variance of a random variable rather than its true variance. The advantage of these bounds is that the quantities they involve are empirical. Previously, these bounds have been applied in sampling procedures [6] and in multi-armed bandit problems [1]. An alternative to empirical risk minimization, called sample variance penalization [5], has been proposed and is motivated by empirical Bernstein bounds.

A new boosting algorithm is proposed in this paper which implements sample variance penalization. The algorithm minimizes the empirical risk on the training set as well as the empirical variance. The two quantities (the risk and the variance) are traded-off through a scalar parameter. Moreover, the

algorithm proposed in this article does not require exhaustive enumeration of the weak learners (unlike an earlier algorithm by [10]).

Assume that a training set $(X_i, y_i)_{i=1}^n$ is provided where $X_i \in \mathcal{X}$ and $y_i \in \{\pm 1\}$ are drawn independently and identically distributed (*iid*) from a fixed but unknown distribution $\mathcal{D}$. The goal is to learn a classifier or a function $f : \mathcal{X} \to \{\pm 1\}$ that performs well on test examples drawn from the same distribution $\mathcal{D}$. In the rest of this article, $G : \mathcal{X} \to \{\pm 1\}$ denotes the so-called weak learner. The notation $G^s$ denotes the weak learner in a particular iteration $s$. Further, the two indices sets $I_s$ and $J_s$, respectively, denote examples that the weak learner $G^s$ correctly classified and misclassified, i.e., $I_s := \{i|G^s(X_i) = y_i\}$ and $J_s := \{j|G^s(X_j) \neq y_j\}$.

---

**Algorithm 1** AdaBoost    Require: $(X_i, y_i)_{i=1}^n$, and weak learners $\mathcal{H}$

---

Initialize the weights: $w_i \leftarrow 1/n$ for $i = 1, \ldots, n$; Initialize $f$ to predict zero on all inputs.
**for** $s \leftarrow 1$ to $S$ **do**
    Estimate a weak learner $G^s(\cdot)$ from training examples weighted by $(w_i)_{i=1}^n$.
    $\alpha_s = \frac{1}{2} \log \left( \sum_{i:G^s(X_i)=y_i} w_i \,/\, \sum_{j:G^s(X_j) \neq y_j} w_j \right)$
    **if** $\alpha_s \leq 0$ **then** break **end if**
    $f(\cdot) \leftarrow f(\cdot) + \alpha_s G^s(\cdot)$
    $w_i \leftarrow w_i \exp(-y_i G^s(X_i) \alpha_s)/Z_s$ where $Z_s$ is such that $\sum_{i=1}^n w_i = 1$.
**end for**

---

---

**Algorithm 2** VadaBoost    Require: $(X_i, y_i)_{i=1}^n$, scalar parameter $1 \geq \lambda \geq 0$, and weak learners $\mathcal{H}$

---

Initialize the weights: $w_i \leftarrow 1/n$ for $i = 1, \ldots, n$; Initialize $f$ to predict zero on all inputs.
**for** $s \leftarrow 1$ to $S$ **do**
    $u_i \leftarrow \lambda n w_i^2 + (1 - \lambda) w_i$
    Estimate a weak learner $G^s(\cdot)$ from training examples weighted by $(u_i)_{i=1}^n$.
    $\alpha_s = \frac{1}{4} \log \left( \sum_{i:G^s(X_i)=y_i} u_i \,/\, \sum_{j:G^s(X_j) \neq y_j} u_j \right)$
    **if** $\alpha_s \leq 0$ **then** break **end if**
    $f(\cdot) \leftarrow f(\cdot) + \alpha_s G^s(\cdot)$
    $w_i \leftarrow w_i \exp(-y_i G^s(X_i) \alpha_s)/Z_s$ where $Z_s$ is such that $\sum_{i=1}^n w_i = 1$.
**end for**

---

## 2    Algorithms

In this section, we briefly discuss AdaBoost [4] and then propose a new algorithm called the VadaBoost. The derivation of VadaBoost will be provided in detail in the next section.

AdaBoost (Algorithm 1) assigns a weight $w_i$ to each training example. In each step of the AdaBoost, a weak learner $G^s(\cdot)$ is obtained on the weighted examples and a weight $\alpha_s$ is assigned to it. Thus, AdaBoost iteratively builds $\sum_{s=1}^S \alpha_s G^s(\cdot)$. If a training example is correctly classified, its weight is exponentially decreased; if it is misclassified, its weight is exponentially increased. The process is repeated until a stopping criterion is met. AdaBoost essentially performs empirical risk minimization: $\min_{f \in \mathcal{F}} \left( \frac{1}{n} \sum_{i=1}^n e^{-y_i f(X_i)} \right)$ by greedily constructing the function $f(\cdot)$ via $\sum_{s=1}^S \alpha_s G^s(\cdot)$.

Recently an alternative to empirical risk minimization has been proposed. This new criterion, known as the sample variance penalization [5] trades-off the empirical risk with the empirical variance:

$$\arg \min_{f \in \mathcal{F}} \frac{1}{n} \sum_{i=1}^n l(f(X_i), y_i) + \tau \sqrt{\frac{\hat{\mathbf{V}}[l(f(X), y)]}{n}}, \tag{1}$$

where $\tau \geq 0$ explores the trade-off between the two quantities. The motivation for sample variance penalization comes from the following theorem [5]:

**Theorem 1** *Let $(X_i, y_i)_{i=1}^n$ be drawn* iid *from a distribution $\mathcal{D}$. Let $\mathcal{F}$ be a class of functions $f : \mathcal{X} \to \mathbf{R}$. Then, for a loss $l : \mathbf{R} \times \mathcal{Y} \to [0, 1]$, for any $\delta > 0$, w.p. at least $1 - \delta$, $\forall f \in \mathcal{F}$*

$$\mathbf{E}[l(f(X), y)] \leq \frac{1}{n} \sum_{i=1}^n l(f(X_i), y_i) + \frac{15 \ln(\mathcal{M}(n)/\delta)}{(n-1)} + \sqrt{\frac{18 \hat{\mathbf{V}}[l(f(X), y)] \ln(\mathcal{M}(n)/\delta)}{n}}, \quad (2)$$

*where $\mathcal{M}(n)$ is a complexity measure.*

From the above uniform convergence result, it can be argued that future loss can be minimized by minimizing the right hand side of the bound on training examples. Since the variance $\hat{\mathbf{V}}[l(f(X), y)]$ has a multiplicative factor involving $\mathcal{M}(n)$, $\delta$ and $n$, for a given problem, it is difficult to specify the relative importance between empirical risk and empirical variance a priori. Hence, sample variance penalization (1) necessarily involves a trade-off parameter $\tau$.

Empirical risk minimization or sample variance penalization on the $0 - 1$ loss is a hard problem; this problem is often circumvented by minimizing a convex upper bound on the $0 - 1$ loss. In this paper, we consider the exponential loss $l(f(X), y) := e^{-yf(X)}$. With the above loss, it was shown by [10] that sample variance penalization is equivalent to minimizing the following cost,

$$\left( \sum_{i=1}^n e^{-y_i f(X_i)} \right)^2 + \lambda \left( n \sum_{i=1}^n e^{-2y_i f(X_i)} - \left( \sum_{i=1}^n e^{-y_i f(X_i)} \right)^2 \right). \quad (3)$$

Theorem 1 requires that the loss function be bounded. Even though the exponential loss is unbounded, boosting is typically performed only for a finite number of iterations in most practical applications. Moreover, since weak learners typically perform only slightly better than random guessing, each $\alpha_s$ in AdaBoost (or in VadaBoost) is typically small thus limiting the range of the function learned. Furthermore, experiments will confirm that sample variance penalization results in a significant empirical performance improvement over empirical risk minimization.

Our proposed algorithm is called VadaBoost[1] and is described in Algorithm 2. VadaBoost iteratively performs sample variance penalization (i.e., it minimizes the cost (3) iteratively). Clearly, VadaBoost shares the simplicity and ease of implementation found in AdaBoost.

## 3 Derivation of VadaBoost

In the $s^{th}$ iteration, our objective is to choose a weak learner $G^s$ and a weight $\alpha_s$ such that $\sum_{t=1}^s \alpha_t G^t(\cdot)$ reduces the cost (3). Denote by $w_i$ the quantity $e^{-y_i \sum_{t=1}^{s-1} \alpha_t G^t(x_i)}/Z_s$. Given a candidate weak learner $G^s(\cdot)$, the cost (3) for the function $\sum_{t=1}^{s-1} \alpha_t G^t(\cdot) + \alpha G^s(\cdot)$ can be expressed as a function of $\alpha$:

$$V(\alpha; \mathbf{w}, \lambda, I, J) :=$$

$$\left( \sum_{i \in I} w_i e^{-\alpha} + \sum_{j \in J} w_j e^{\alpha} \right)^2 + \lambda \left( n \sum_{i \in I} w_i^2 e^{-2\alpha} + n \sum_{j \in J} w_j^2 e^{2\alpha} - \left( \sum_{i \in I} w_i e^{-\alpha} + \sum_{j \in J} w_j e^{\alpha} \right)^2 \right). \quad (4)$$

up to a multiplicative factor. In the quantity above, $I$ and $J$ are the two index sets (of correctly classified and incorrectly classified examples) over $G^s$. Let the vector $\mathbf{w}$ whose $i^{th}$ component is $w_i$ denote the current set of weights on the training examples. Here, we have dropped the subscripts/superscripts $s$ for brevity.

**Lemma 2** *The update of $\alpha_s$ in Algorithm 2 minimizes the cost*

$$U(\alpha; \mathbf{w}, \lambda, I, J) := \left( \sum_{i \in I} \left( \lambda n w_i^2 + (1 - \lambda) w_i \right) \right) e^{-2\alpha} + \left( \sum_{j \in J} \left( \lambda n w_j^2 + (1 - \lambda) w_j \right) \right) e^{2\alpha}. \quad (5)$$

**Proof** By obtaining the second derivative of the above expression (with respect to $\alpha$), it is easy to see that it is convex in $\alpha$. Thus, setting the derivative with respect to $\alpha$ to zero gives the optimal choice of $\alpha$ as shown in Algorithm 2. ∎

**Theorem 3** *Assume that $0 \leq \lambda \leq 1$ and $\sum_{i=1}^{n} w_i = 1$ (i.e. normalized weights). Then, $V(\alpha; \mathbf{w}, \lambda, I, J) \leq U(\alpha; \mathbf{w}, \lambda, I, J)$ and $V(0; \mathbf{w}, \lambda, I, J) = U(0; \mathbf{w}, \lambda, I, J)$. That is, U is an upper bound on V and the bound is exact at $\alpha = 0$.*

**Proof** Denoting $1 - \lambda$ by $\bar{\lambda}$, we have:

$$
V(\alpha; \mathbf{w}, \lambda, I, J) = \left( \sum_{i \in I} w_i e^{-\alpha} + \sum_{j \in J} w_j e^{\alpha} \right)^2 + \lambda \left( n \sum_{i \in I} w_i^2 e^{-2\alpha} + n \sum_{j \in J} w_j^2 e^{2\alpha} - \left( \sum_{i \in I} w_i e^{-\alpha} + \sum_{j \in J} w_j e^{\alpha} \right)^2 \right)
$$

$$
= \bar{\lambda} \left( \sum_{i \in I} w_i e^{-\alpha} + \sum_{j \in J} w_j e^{\alpha} \right)^2 + \lambda \left( n \sum_{i \in I} w_i^2 e^{-2\alpha} + n \sum_{j \in J} w_j^2 e^{2\alpha} \right)
$$

$$
= \lambda \left( n \sum_{i \in I} w_i^2 e^{-2\alpha} + n \sum_{j \in J} w_j^2 e^{2\alpha} \right) + \bar{\lambda} \left( \left( \sum_{i \in I} w_i \right)^2 e^{-2\alpha} + \left( \sum_{j \in J} w_j \right)^2 e^{2\alpha} + 2 \left( \sum_{i \in I} w_i \right) \left( \sum_{j \in J} w_j \right) \right)
$$

$$
= \lambda \left( n \sum_{i \in I} w_i^2 e^{-2\alpha} + n \sum_{j \in J} w_j^2 e^{2\alpha} \right) + \bar{\lambda} \left( \left( \sum_{i \in I} w_i \right) \left( 1 - \sum_{j \in J} w_j \right) e^{-2\alpha} \right.
$$

$$
\left. + \left( \sum_{j \in J} w_j \right) \left( 1 - \sum_{i \in I} w_i \right) e^{2\alpha} + 2\bar{\lambda} \left( \sum_{i \in I} w_i \right) \left( \sum_{j \in J} w_j \right) \right)
$$

$$
= \left( \sum_{i \in I} \left( \lambda n w_i^2 + \bar{\lambda} w_i \right) \right) e^{-2\alpha} + \left( \sum_{j \in J} \left( \lambda n w_j^2 + \bar{\lambda} w_j \right) \right) e^{2\alpha}
$$

$$
+ \bar{\lambda} \left( \sum_{i \in I} w_i \right) \left( \sum_{j \in J} w_j \right) \left( -e^{2\alpha} - e^{-2\alpha} + 2 \right)
$$

$$
\leq \left( \sum_{i \in I} \left( \lambda n w_i^2 + \bar{\lambda} w_i \right) \right) e^{-2\alpha} + \left( \sum_{j \in J} \left( \lambda n w_j^2 + \bar{\lambda} w_j \right) \right) e^{2\alpha} = U(\alpha; \mathbf{w}, \lambda, I, J).
$$

On line two, terms were simply regrouped. On line three, the square term from line two was expanded. On the next line, we used the fact that $\sum_{i \in I} w_i + \sum_{j \in J} w_j = \sum_{i=1}^{n} w_i = 1$. On the fifth line, we once again regrouped terms; the last term in this expression (which is $e^{2\alpha} + e^{-2\alpha} - 2$) can be written as $(e^{\alpha} - e^{-\alpha})^2$. When $\alpha = 0$ this term vanishes. Hence the bound is exact at $\alpha = 0$. ∎

**Corollary 4** *VadaBoost monotonically decreases the cost* (3).

The above corollary follows from:

$$
V(\alpha_s; \mathbf{w}, \lambda, I, J) \leq U(\alpha_s; \mathbf{w}, \lambda, I, J) < U(0; \mathbf{w}, \lambda, I, J) = V(0; \mathbf{w}, \lambda, I, J).
$$

In the above, the first inequality follows from Theorem (3). The second strict inequality holds because $\alpha_s$ is a minimizer of $U$ from Lemma (2); it is not hard to show that $U(\alpha_s; \mathbf{w}, \lambda, I, J)$ is strictly less than $U(0; \mathbf{w}, \lambda, I, J)$ from the termination criterion of VadaBoost. The third equality again follows from Theorem (3). Finally, we notice that $V(0; \mathbf{w}, \lambda, I, J)$ merely corresponds to the cost (3) at $\sum_{t=1}^{s-1} \alpha_t G^t(\cdot)$. Thus, we have shown that taking a step $\alpha_s$ decreases the cost (3).

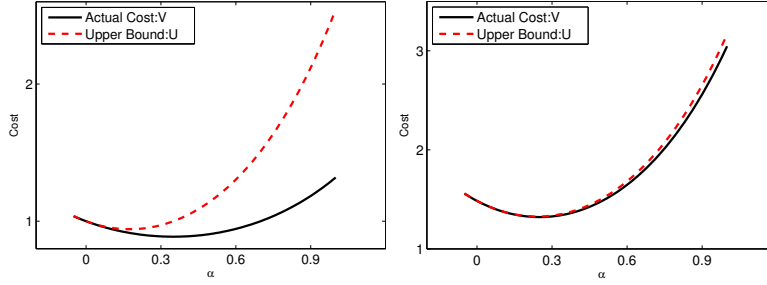

Figure 1: Typical Upper bound $U(\alpha; \mathbf{w}, \lambda, I, J)$ and the actual cost function $V(\alpha; \mathbf{w}, \lambda, I, J)$ values under varying $\alpha$. The bound is exact at $\alpha = 0$. The bound gets closer to the actual function value as $\lambda$ grows. The left plot shows the bound for $\lambda = 0$ and the right plot shows it for $\lambda = 0.9$

We point out that we use a different upper bound in each iteration since $V$ and $U$ are parameterized by the current weights in the VadaBoost algorithm. Also note that our upper bound holds only for $0 \leq \lambda \leq 1$. Although the choice $0 \leq \lambda \leq 1$ seems restrictive, intuitively, it is natural to have a higher penalization on the empirical mean rather than the empirical variance during minimization. Also, a closer look at the empirical Bernstein inequality in [5] shows that the empirical variance term is multiplied by $\sqrt{1/n}$ while the empirical mean is multiplied by one. Thus, for large values of $n$, the weight on the sample variance is small. Furthermore, our experiments suggest that restricting $\lambda$ to this range does not significantly change the results.

## 4 How good is the upper bound?

First, we observe that our upper bound is exact when $\lambda = 1$. Also, our upper bound is loosest for the case $\lambda = 0$. We visualize the upper bound and the true cost for two settings of $\lambda$ in Figure 1.

Since the cost (4) is minimized via an upper bound (5), a natural question is: how good is this approximation? We evaluate the tightness of this upper bound by considering its impact on learning efficiency. As is clear from figure (1), when $\lambda = 1$, the upper bound is exact and incurs no inefficiency. In the other extreme when $\lambda = 0$, the cost of VadaBoost coincides with AdaBoost and the bound is effectively at its loosest. Even in this extreme case, VadaBoost derived through an upper bound only requires at most twice the number of iterations as AdaBoost to achieve a particular cost. The following theorem shows that our algorithm remains efficient even in this worst-case scenario.

**Theorem 5** *Let $O_A$ denote the squared cost obtained by AdaBoost after $S$ iterations. For weak learners in any iteration achieving a fixed error rate $\epsilon < 0.5$, VadaBoost with the setting $\lambda = 0$ attains a cost at least as low as $O_A$ in no more than $2S$ iterations.*

**Proof** Denote the weight on the example $i$ in $s^{\text{th}}$ iteration by $w_i^s$. The weighted error rate of the $s^{\text{th}}$ classifier is $\epsilon_s = \sum_{j \in J_s} w_j^s$. We have, for both algorithms,

$$w_i^{S+1} = \frac{w_i^S \exp(-y_i \alpha_S G^S(X_i))}{Z_s} = \frac{\exp(-y_i \sum_{s=1}^{S} \alpha_s G^s(X_i))}{n \prod_{s=1}^{S} Z_s}. \tag{6}$$

The value of the normalization factor in the case of AdaBoost is

$$Z_s^a = \sum_{j \in j_s} w_j^s e^{\alpha_s} + \sum_{i \in I_s} w_i^s e^{-\alpha_s} = 2\sqrt{\epsilon_s(1 - \epsilon_s)}. \tag{7}$$

Similarly, the value of the normalization factor for VadaBoost is given by

$$Z_s^v = \sum_{j \in J_s} w_j^s e^{\alpha_s} + \sum_{i \in I_s} w_i^s e^{-\alpha_s} = ((\epsilon_s)(1 - \epsilon_s))^{\frac{1}{4}} (\sqrt{\epsilon_s} + \sqrt{1 - \epsilon_s}). \tag{8}$$

The squared cost function of AdaBoost after $S$ steps is given by

$$O_A = \left( \sum_{i=1}^n \exp(-y_i \sum_{s=1}^S \alpha_s y_i G^s(X)) \right)^2 = \left( n \prod_{s=1}^S Z_s^a \sum_{i=1}^n w_i^{s+1} \right)^2 = n^2 \left( \prod_{s=1}^S Z_s^a \right)^2 = n^2 \prod_{s=1}^S 4\epsilon_s(1 - \epsilon_s).$$

We used (6), (7) and the fact that $\sum_{i=1}^n w_i^{S+1} = 1$ to derive the above expression. Similarly, for $\lambda = 0$ the cost of VadaBoost satisfies[2]

$$O_V = \left( \sum_{i=1}^n \exp(-y_i \sum_{s=1}^S \alpha_s y_i G^s(X)) \right)^2 = \left( n \prod_{s=1}^S Z_s^a \sum_{i=1}^n w_i^{s+1} \right)^2 = n^2 \left( \prod_{s=1}^S Z_s^v \right)^2$$

$$= n^2 \prod_{s=1}^S (2\epsilon_s(1 - \epsilon_s) + \sqrt{\epsilon_s(1 - \epsilon_s)}).$$

Now, suppose that $\epsilon_s = \epsilon$ for all $s$. Then, the squared cost achieved by AdaBoost is given by $n^2(4\epsilon(1 - \epsilon))^S$. To achieve the same cost value, VadaBoost, with weak learners with the same error rate needs at most $S \frac{\log(4\epsilon(1-\epsilon))}{\log(2\epsilon(1-\epsilon)+\sqrt{\epsilon(1-\epsilon)})}$ times. Within the range of interest for $\epsilon$, the term multiplying $S$ above is at most 2. ∎

Although the above worse-case bound achieves a factor of two, for $\epsilon > 0.4$, VadaBoost requires only about 33% more iterations than AdaBoost. To summarize, even in the worst possible scenario where $\lambda = 0$ (when the variational bound is at its loosest), the VadaBoost algorithm takes no more than double (a small constant factor) the number of iterations of AdaBoost to achieve the same cost.

---

**Algorithm 3** EBBoost:    Require: $(X_i, y_i)_{i=1}^n$, scalar parameter $\lambda \geq 0$, and weak learners $\mathcal{H}$

---

Initialize the weights: $w_i \leftarrow 1/n$ for $i = 1, \ldots, n$; Initialize $f$ to predict zero on all inputs.
**for** $s \leftarrow 1$ to $S$ **do**
    Get a weak learner $G^s(\cdot)$ that minimizes (3) with the following choice of $\alpha_s$:
    $\alpha_s = \frac{1}{4} \log \left( \frac{(1-\lambda)(\sum_{i \in I_s} w_i)^2 + \lambda n \sum_{i \in I_s} w_i^2}{(1-\lambda)(\sum_{i \in J_s} w_i)^2 + \lambda n \sum_{i \in J_s} w_i^2} \right)$
    **if** $\alpha_s < 0$ **then** break **end if**
    $f(\cdot) \leftarrow f(\cdot) + \alpha_s G^s(\cdot)$
    $w_i \leftarrow w_i \exp(-y_i G^s(X_i) \alpha_s)/Z_s$ where $Z_s$ is such that $\sum_{i=1}^n w_i = 1$.
**end for**

---

## 5 A limitation of the EBBoost algorithm

A sample variance penalization algorithm known as EBBoost was previously explored [10]. While this algorithm was simple to implement and showed significant improvements over AdaBoost, it suffers from a severe limitation: it requires enumeration and evaluation of every possible weak learner per iteration. Recall the steps implementing EBBoost in Algorithm 3. An implementation of EBBoost requires exhaustive enumeration of weak learners in search of the one that minimizes cost (3). It is preferable, instead, to find the best weak learner by providing weights on the training examples and efficiently computing the rule whose performance on that weighted set of examples is guaranteed to be better than random guessing. However, with the EBBoost algorithm, the weight on all the misclassified examples is $\sum_{i \in J_s} w_i^2 + \left( \sum_{i \in J_s} w_i \right)^2$ and the weight on correctly classified examples is $\sum_{i \in I_s} w_i^2 + \left( \sum_{i \in I_s} w_i \right)^2$; these aggregate weights on misclassified examples and correctly classified examples do not translate into weights on the individual examples. Thus, it becomes necessary to exhaustively enumerate weak learners in Algorithm 3. While enumeration of weak learners is possible in the case of decision stumps, it poses serious difficulties in the case of weak learners such as decision trees, ridge regression, etc. Thus, VadaBoost is the more versatile boosting algorithm for sample variance penalization.

Table 1: Mean and standard errors with decision stump as the weak learner.

| Dataset | AdaBoost | EBBoost | VadaBoost | RLP-Boost | RQP-Boost |
|---|---|---|---|---|---|
| a5a | $16.15 \pm 0.1$ | $16.05 \pm 0.1$ | $16.22 \pm 0.1$ | $16.21 \pm 0.1$ | $16.04 \pm 0.1$ |
| abalone | $21.64 \pm 0.2$ | $21.52 \pm 0.2$ | $21.63 \pm 0.2$ | $22.29 \pm 0.2$ | $21.79 \pm 0.2$ |
| image | $3.37 \pm 0.1$ | $3.14 \pm 0.1$ | $3.14 \pm 0.1$ | $3.18 \pm 0.1$ | $3.09 \pm 0.1$ |
| mushrooms | $0.02 \pm 0.0$ | $0.02 \pm 0.0$ | $0.01 \pm 0.0$ | $0.01 \pm 0.0$ | $0.00 \pm 0.0$ |
| musk | $3.84 \pm 0.1$ | $3.51 \pm 0.1$ | $3.59 \pm 0.1$ | $3.60 \pm 0.1$ | $3.41 \pm 0.1$ |
| mnist09 | $0.89 \pm 0.0$ | $0.85 \pm 0.0$ | $0.84 \pm 0.0$ | $0.98 \pm 0.0$ | $0.88 \pm 0.0$ |
| mnist14 | $0.64 \pm 0.0$ | $0.58 \pm 0.0$ | $0.60 \pm 0.0$ | $0.68 \pm 0.0$ | $0.63 \pm 0.0$ |
| mnist27 | $2.11 \pm 0.1$ | $1.86 \pm 0.1$ | $2.01 \pm 0.1$ | $2.06 \pm 0.1$ | $1.95 \pm 0.1$ |
| mnist38 | $4.45 \pm 0.1$ | $4.12 \pm 0.1$ | $4.32 \pm 0.1$ | $4.51 \pm 0.1$ | $4.25 \pm 0.1$ |
| mnist56 | $2.79 \pm 0.1$ | $2.56 \pm 0.1$ | $2.62 \pm 0.1$ | $2.77 \pm 0.1$ | $2.72 \pm 0.1$ |
| ringnorm | $13.16 \pm 0.6$ | $11.74 \pm 0.6$ | $12.46 \pm 0.6$ | $13.02 \pm 0.6$ | $12.86 \pm 0.6$ |
| spambase | $5.90 \pm 0.1$ | $5.64 \pm 0.1$ | $5.78 \pm 0.1$ | $5.81 \pm 0.1$ | $5.75 \pm 0.1$ |
| splice | $8.83 \pm 0.2$ | $8.33 \pm 0.1$ | $8.48 \pm 0.1$ | $8.55 \pm 0.2$ | $8.47 \pm 0.1$ |
| twonorm | $3.16 \pm 0.1$ | $2.98 \pm 0.1$ | $3.09 \pm 0.1$ | $3.29 \pm 0.1$ | $3.07 \pm 0.1$ |
| w4a | $2.60 \pm 0.1$ | $2.38 \pm 0.1$ | $2.50 \pm 0.1$ | $2.44 \pm 0.1$ | $2.36 \pm 0.1$ |
| waveform | $10.99 \pm 0.1$ | $10.96 \pm 0.1$ | $10.75 \pm 0.1$ | $10.95 \pm 0.1$ | $10.60 \pm 0.1$ |
| wine | $23.62 \pm 0.2$ | $23.52 \pm 0.2$ | $23.41 \pm 0.1$ | $24.16 \pm 0.1$ | $23.61 \pm 0.1$ |
| wisc | $5.32 \pm 0.3$ | $4.38 \pm 0.2$ | $5.00 \pm 0.2$ | $4.96 \pm 0.3$ | $4.72 \pm 0.3$ |

Table 2: Mean and standard errors with CART as the weak learner.

| Dataset | AdaBoost | VadaBoost | RLP-Boost | RQP-Boost |
|---|---|---|---|---|
| a5a | $17.59 \pm 0.2$ | $17.16 \pm 0.1$ | $18.24 \pm 0.1$ | $17.99 \pm 0.1$ |
| abalone | $21.87 \pm 0.2$ | $21.30 \pm 0.2$ | $22.16 \pm 0.2$ | $21.84 \pm 0.2$ |
| image | $1.93 \pm 0.1$ | $1.98 \pm 0.1$ | $1.99 \pm 0.1$ | $1.95 \pm 0.1$ |
| mushrooms | $0.01 \pm 0.0$ | $0.01 \pm 0.0$ | $0.02 \pm 0.0$ | $0.01 \pm 0.0$ |
| musk | $2.36 \pm 0.1$ | $2.07 \pm 0.1$ | $2.40 \pm 0.1$ | $2.29 \pm 0.1$ |
| mnist09 | $0.73 \pm 0.0$ | $0.72 \pm 0.0$ | $0.76 \pm 0.0$ | $0.71 \pm 0.0$ |
| mnist14 | $0.52 \pm 0.0$ | $0.50 \pm 0.0$ | $0.55 \pm 0.0$ | $0.52 \pm 0.0$ |
| mnist27 | $1.31 \pm 0.0$ | $1.24 \pm 0.0$ | $1.32 \pm 0.0$ | $1.29 \pm 0.0$ |
| mnist38 | $1.89 \pm 0.1$ | $1.72 \pm 0.1$ | $1.88 \pm 0.1$ | $1.87 \pm 0.1$ |
| mnist56 | $1.23 \pm 0.1$ | $1.17 \pm 0.0$ | $1.20 \pm 0.0$ | $1.19 \pm 0.1$ |
| ringnorm | $7.94 \pm 0.4$ | $7.78 \pm 0.4$ | $8.60 \pm 0.5$ | $7.84 \pm 0.4$ |
| spambase | $6.14 \pm 0.1$ | $5.76 \pm 0.1$ | $6.25 \pm 0.1$ | $6.03 \pm 0.1$ |
| splice | $4.02 \pm 0.1$ | $3.67 \pm 0.1$ | $4.03 \pm 0.1$ | $3.97 \pm 0.1$ |
| twonorm | $3.40 \pm 0.1$ | $3.27 \pm 0.1$ | $3.50 \pm 0.1$ | $3.38 \pm 0.1$ |
| w4a | $2.90 \pm 0.1$ | $2.90 \pm 0.1$ | $2.90 \pm 0.1$ | $2.90 \pm 0.1$ |
| waveform | $11.09 \pm 0.1$ | $10.59 \pm 0.1$ | $11.11 \pm 0.1$ | $10.82 \pm 0.1$ |
| wine | $21.94 \pm 0.2$ | $21.18 \pm 0.2$ | $22.44 \pm 0.2$ | $22.18 \pm 0.2$ |
| wisc | $4.61 \pm 0.2$ | $4.18 \pm 0.2$ | $4.63 \pm 0.2$ | $4.37 \pm 0.2$ |

## 6   Experiments

In this section, we evaluate the empirical performance of the VadaBoost algorithm with respect to several other algorithms. The primary purpose of our experiments is to compare sample variance penalization versus empirical risk minimization and to show that we can efficiently perform sample variance penalization for weak learners beyond decision stumps. We compared VadaBoost against EBBoost, AdaBoost, regularized LP and QP boost algorithms [7]. All the algorithms except AdaBoost have one extra parameter to tune.

Experiments were performed on benchmark datasets that have been previously used in [10]. These datasets include a variety of tasks including all digits from the MNIST dataset. Each dataset was divided into three parts: 50% for training, 25% for validation and 25% for test. The total number of examples was restricted to 5000 in the case of MNIST and musk datasets due to computational restrictions of solving LP/QP.

The first set of experiments use decision stumps as the weak learners. The second set of experiments used Classification and Regression Trees or CART [3] as weak learners. A standard MATLAB implementation of CART was used without modification. For all the datasets, in both experiments,

AdaBoost, VadaBoost and EBBoost (in the case of stumps) were run until there was no drop in the error rate on the validation for the next 100 consecutive iterations. The values of the parameters for VadaBoost and EBBoost were chosen to minimize the validation error upon termination. RLP-Boost and RQP-Boost were given the predictions obtained by AdaBoost. Their regularization parameter was also chosen to minimize the error rate on the validation set. Once the parameter values were fixed via the validation set, we noted the test set error corresponding to that parameter value. The entire experiment was repeated 50 times by randomly selecting train, test and validation sets. The numbers reported here are average from these runs.

The results for the decision stump and CART experiments are reported in Tables 1 and 2. For each dataset, the algorithm with the best percentage test error is represented by a dark shaded cell. All lightly shaded entries in a row denote results that are not significantly different from the minimum error (according to a paired t-test at a $1\%$ significance level). With decision stumps, both EBBoost and VadaBoost have comparable performance and significantly outperform AdaBoost. With CART as the weak learner, VadaBoost is once again significantly better than AdaBoost.

We gave a guarantee on the number of iterations required in the worst case for Vadaboost (which approximately matches the AdaBoost cost (squared) in Theorem 5). An assumption in that theorem was that the error rate of each weak learner was fixed. However, in practice, the error rates of the weak learners are not constant over the iterations. To see this behavior in practice, we have shown the results with the MNIST 3 versus 8 classification experiment. In figure 2 we show the cost (plus 1) for each algorithm (the AdaBoost cost has been squared) versus the number of iterations using a logarithmic scale on the Y-axis. Since at $\lambda = 0$, EBBoost reduces to AdaBoost, we omit its plot at that setting. From the figure, it can be seen

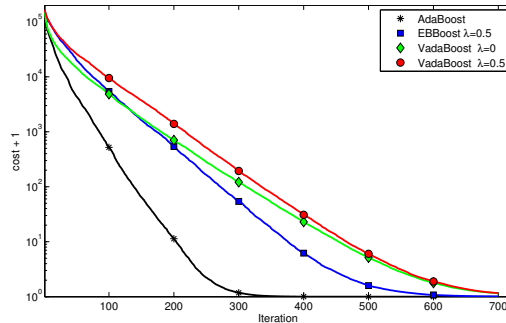

Figure 2: 1+ cost vs the number of iterations.

that the number of iterations required by VadaBoost is roughly twice the number of iterations required by AdaBoost. At $\lambda = 0.5$, there is only a minor difference in the number of iterations required by EBBoost and VadaBoost.

## 7  Conclusions

This paper identified a key weakness in the EBBoost algorithm and proposed a novel algorithm that efficiently overcomes the limitation to enumerable weak learners. VadaBoost reduces a well motivated cost by iteratively minimizing an upper bound which, unlike EBBoost, allows the boosting method to handle any weak learner by estimating weights on the data. The update rule of VadaBoost has a simplicity that is reminiscent of AdaBoost. Furthermore, despite the use of an upper bound, the novel boosting method remains efficient. Even when the bound is at its loosest, the number of iterations required by VadaBoost is a small constant factor more than the number of iterations required by AdaBoost. Experimental results showed that VadaBoost outperforms AdaBoost in terms of classification accuracy and efficiently applying to any family of weak learners. The effectiveness of boosting has been explained via margin theory [9] though it has taken a number of years to settle certain open questions [8]. Considering the simplicity and effectiveness of VadaBoost, one natural future research direction is to study the margin distributions it obtains. Another future research direction is to design efficient sample variance penalization algorithms for other problems such as multi-class classification, ranking, and so on.

**Acknowledgements**   This material is based upon work supported by the National Science Foundation under Grant No. 1117631, by a Google Research Award, and by the Department of Homeland Security under Grant No. N66001-09-C-0080.

## Footnotes

[1]The V in VadaBoost emphasizes the fact that Algorithm 2 penalizes the empirical variance.

[2]The cost which VadaBoost minimizes at $\lambda = 0$ is the squared cost of AdaBoost, we do not square it again.

# References

[1] J-Y. Audibert, R. Munos, and C. Szepesvári. Tuning bandit algorithms in stochastic environments. In *ALT*, 2007.

[2] P. L. Bartlett, M. I. Jordan, and J. D. McAuliffe. Convexity, classification, and risk bounds. *Journal of the American Statistical Association*, 101(473):138–156, 2006.

[3] L. Breiman, J.H. Friedman, R.A. Olshen, and C.J. Stone. *Classification and Regression Trees*. Chapman and Hall, New York, 1984.

[4] Y. Freund and R. E. Schapire. A decision-theoretic generalization of on-line learning and an application to boosting. *Journal of Computer and System Sciences*, 55(1):119–139, 1997.

[5] A. Maurer and M. Pontil. Empirical Bernstein bounds and sample variance penalization. In *COLT*, 2009.

[6] V. Mnih, C. Szepesvári, and J-Y. Audibert. Empirical Bernstein stopping. In *COLT*, 2008.

[7] G. Raetsch, T. Onoda, and K.-R. Muller. Soft margins for AdaBoost. *Machine Learning*, 43:287–320, 2001.

[8] L. Reyzin and R. Schapire. How boosting the margin can also boost classifier complexity. In *ICML*, 2006.

[9] R. E. Schapire, Y. Freund, P. L. Bartlett, and W. S. Lee. Boosting the margin: a new explanation for the effectiveness of voting methods. *Annals of Statistics*, 26(5):1651–1686, 1998.

[10] P. K. Shivaswamy and T. Jebara. Empirical Bernstein boosting. In *AISTATS*, 2010.

[11] V. Vapnik. *The Nature of Statistical Learning Theory*. Springer, New York, NY, 1995.

